# Automatic choice of dimensionality for PCA

**Thomas P. Minka**
MIT Media Lab
20 Ames St, Cambridge, MA 02139
tpminka@media.mit.edu

## Abstract

A central issue in principal component analysis (PCA) is choosing the number of principal components to be retained. By interpreting PCA as density estimation, we show how to use Bayesian model selection to estimate the true dimensionality of the data. The resulting estimate is simple to compute yet guaranteed to pick the correct dimensionality, given enough data. The estimate involves an integral over the Steifel manifold of $k$-frames, which is difficult to compute exactly. But after choosing an appropriate parameterization and applying Laplace's method, an accurate and practical estimator is obtained. In simulations, it is convincingly better than cross-validation and other proposed algorithms, plus it runs much faster.

## 1 Introduction

Recovering the intrinsic dimensionality of a data set is a classic and fundamental problem in data analysis. A popular method for doing this is PCA or localized PCA. Modeling the data manifold with localized PCA dates back to [4]. Since then, the problem of spacing and sizing the local regions has been solved via the EM algorithm and split/merge techniques [2, 6, 14, 5].

However, the task of dimensionality selection has not been solved in a satisfactory way. On the one hand we have crude methods based on eigenvalue thresholding [4] which are very fast, or we have iterative methods [1] which require excessive computing time. This paper resolves the situation by deriving a method which is both accurate and fast. It is an application of Bayesian model selection to the probabilistic PCA model developed by [12, 15].

The new method operates exclusively on the eigenvalues of the data covariance matrix. In the local PCA context, these would be the eigenvalues of the local responsibility-weighted covariance matrix, as defined by [14]. The method can be used to fit different PCA models to different classes, for use in Bayesian classification [11].

## 2 Probabilistic PCA

This section reviews the results of [15]. The PCA model is that a $d$-dimensional vector $\mathbf{x}$ was generated from a smaller $k$-dimensional vector $\mathbf{w}$ by a linear transformation $(\mathbf{H}, \mathbf{m})$

plus a noise vector $\mathbf{e}$: $\mathbf{x} = \mathbf{Hw} + \mathbf{m} + \mathbf{e}$. Both the noise and the principal component vector $\mathbf{w}$ are assumed spherical Gaussian:

$$p(\mathbf{e}) \sim \mathcal{N}(\mathbf{0}, v\mathbf{I}_d) \qquad p(\mathbf{w}) \sim \mathcal{N}(\mathbf{0}, \mathbf{I}_k) \tag{1}$$

The observation $\mathbf{x}$ is therefore Gaussian itself:

$$p(\mathbf{x}|\mathbf{H}, \mathbf{m}, v) \sim \mathcal{N}(\mathbf{m}, \mathbf{HH}^{\mathrm{T}} + v\mathbf{I}) \tag{2}$$

The goal of PCA is to estimate the basis vectors $\mathbf{H}$ and the noise variance $v$ from a data set $D = \{\mathbf{x}_1, ..., \mathbf{x}_N\}$. The probability of the data set is

$$p(D|\mathbf{H}, \mathbf{m}, v) = (2\pi)^{-Nd/2} \left|\mathbf{HH}^{\mathrm{T}} + v\mathbf{I}\right|^{-N/2} \exp(-\frac{1}{2}\mathrm{tr}((\mathbf{HH}^{\mathrm{T}} + v\mathbf{I})^{-1}\mathbf{S})) \tag{3}$$

$$\mathbf{S} = \sum_i (\mathbf{x}_i - \mathbf{m})(\mathbf{x}_i - \mathbf{m})^{\mathrm{T}} \tag{4}$$

As shown by [15], the maximum-likelihood estimates are:

$$\hat{\mathbf{m}} = \frac{1}{N}\sum_i \mathbf{x}_i \qquad \hat{v} = \frac{\sum_{j=k+1}^d \lambda_j}{d-k} \qquad \hat{\mathbf{H}} = \mathbf{U}(\mathbf{\Lambda} - \hat{v}\mathbf{I}_k)^{1/2}\mathbf{R} \tag{5}$$

where orthogonal matrix $\mathbf{U}$ contains the top $k$ eigenvectors of $\mathbf{S}/N$, diagonal matrix $\mathbf{\Lambda}$ contains the corresponding eigenvalues, and $\mathbf{R}$ is an arbitrary orthogonal matrix.

## 3  Bayesian model selection

Bayesian model selection scores models according to the probability they assign the observed data [9, 8]. It is completely analogous to Bayesian classification. It automatically encodes a preference for simpler, more constrained models, as illustrated in figure 1. Simple models only fit a small fraction of data sets, but they assign correspondingly higher probability to those data sets. Flexible models spread themselves out more thinly.

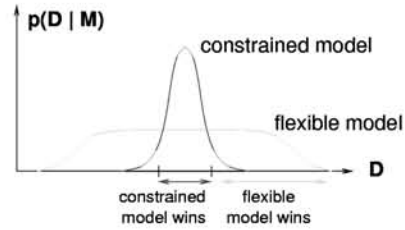

Figure 1: Why Bayesian model selection prefers simpler models

The probability of the data given the model is computed by integrating over the unknown parameter values in that model:

$$p(D|M) = \int_\theta p(D|\theta)p(\theta|M)d\theta \tag{6}$$

This quantity is called the evidence for model $M$. A useful property of Bayesian model selection is that it is guaranteed to select the true model, if it is among the candidates, as the size of the dataset grows to infinity.

### 3.1  The evidence for probabilistic PCA

For the PCA model, we want to select the subspace dimensionality $k$. To do this, we compute the probability of the data for each possible dimensionality and pick the maximum. For a given dimensionality, this requires integrating over all PCA parameters $(\mathbf{m}, \mathbf{H}, v)$. First we need to define a prior density for these parameters. Assuming there is no information

other than the data $D$, the prior should be as noninformative as possible. A noninformative prior for $\mathbf{m}$ is uniform, and with such a prior we can integrate out $\mathbf{m}$ analytically, leaving

$$p(D|\mathbf{H}, v) = N^{-d/2}(2\pi)^{-(N-1)d/2} \left|\mathbf{HH}^\mathrm{T} + v\mathbf{I}\right|^{-(N-1)/2} \exp(-\frac{1}{2}\mathrm{tr}((\mathbf{HH}^\mathrm{T}+v\mathbf{I})^{-1}\mathbf{S}))$$
(7)

$$\text{where } \mathbf{S} = \sum_i (\mathbf{x}_i - \hat{\mathbf{m}})(\mathbf{x}_i - \hat{\mathbf{m}})^\mathrm{T}$$
(8)

Unlike $\mathbf{m}$, $\mathbf{H}$ must have a proper prior since it varies in dimension for different models. Let $\mathbf{H}$ be decomposed just as in (5):

$$\mathbf{H} = \mathbf{U}(\mathbf{L} - v\mathbf{I}_k)^{1/2}\mathbf{R} \qquad \mathbf{U}^\mathrm{T}\mathbf{U} = \mathbf{I}_k \qquad \mathbf{R}^\mathrm{T}\mathbf{R} = \mathbf{I}_k$$
(9)

where $\mathbf{L}$ is diagonal with diagonal elements $l_i$. The orthogonal matrix $\mathbf{U}$ is the basis, $\mathbf{L}$ is the scaling (corrected for noise), and $\mathbf{R}$ is a rotation within the subspace (which will turn out to be irrelevant). A conjugate prior for $(\mathbf{U}, \mathbf{L}, \mathbf{R}, v)$, parameterized by $\alpha$, is

$$p(\mathbf{U}, \mathbf{L}, \mathbf{R}, v) \quad \propto \quad \left|\mathbf{HH}^\mathrm{T} + v\mathbf{I}\right|^{-(\alpha+2)/2} \exp(-\frac{\alpha}{2}\mathrm{tr}((\mathbf{HH}^\mathrm{T} + v\mathbf{I})^{-1}))$$
(10)

This distribution happens to factor into $p(\mathbf{U})p(\mathbf{L})p(\mathbf{R})p(v)$, which means the variables are a-priori independent:

$$p(\mathbf{L}) \quad \propto \quad |\mathbf{L}|^{-(\alpha+2)/2} \exp(-\frac{\alpha}{2}\mathrm{tr}(\mathbf{L}^{-1}))$$
(11)

$$p(v) \quad \propto \quad v^{-(\alpha+2)(d-k)/2} \exp(-\frac{\alpha(d-k)}{2v})$$
(12)

$$p(\mathbf{U})p(\mathbf{R}) \quad = \quad (\text{constant—defined in (20)})$$
(13)

The hyperparameter $\alpha$ controls the sharpness of the prior. For a noninformative prior, $\alpha$ should be small, making the prior diffuse. Besides providing a convenient prior, the decomposition (9) is important for removing redundant degrees of freedom ($\mathbf{R}$) and for separating $\mathbf{H}$ into independent components, as described in the next section.

Combining the likelihood with the prior gives

$$p(D|k) = c_k \int \left|\mathbf{HH}^\mathrm{T} + v\mathbf{I}\right|^{-n/2} \exp(-\frac{1}{2}\mathrm{tr}((\mathbf{HH}^\mathrm{T} + v\mathbf{I})^{-1}(\mathbf{S} + \alpha\mathbf{I}))) \, d\mathbf{U}d\mathbf{L}dv$$
(14)

$$n = N + 1 + \alpha$$
(15)

The constant $c_k$ includes $N^{-d/2}$ and the normalizing terms for $p(\mathbf{U})$, $p(\mathbf{L})$, and $p(v)$ (given in [10])—only $p(\mathbf{U})$ will matter in the end. In this formula $\mathbf{R}$ has already been integrated out; the likelihood does not involve $\mathbf{R}$ so we just get a multiplicative factor of $\int_\mathbf{R} p(\mathbf{R}) \, d\mathbf{R} = 1$.

## 3.2 Laplace approximation

Laplace's method is a powerful method for approximating integrals in Bayesian statistics [8]:

$$\int f(\theta)d\theta \quad \approx \quad f(\hat{\theta})(2\pi)^{\mathrm{rows}(A)/2} \left|\mathbf{A}\right|^{-1/2}$$
(16)

$$\hat{\theta} = \overset{\mathrm{argmax}}{\theta} f(\theta) \qquad \mathbf{A} = -\left[\frac{d^2 \log f(\theta)}{d\theta_i d\theta_j}\right]_{\theta = \hat{\theta}}$$
(17)

The key to getting a good approximation is choosing a good parameterization for $\theta = (\mathbf{U}, \mathbf{L}, v)$. Since $l_i$ and $v$ are positive scale parameters, it is best to use $l_i' = \log(l_i)$ and

$v' = \log(v)$. This results in

$$\hat{l}_i = \frac{N\lambda_i + \alpha}{N - 1 + \alpha} \qquad \hat{v} = \frac{N\sum_{j=k+1}^d \lambda_j}{n(d-k) - 2} \tag{18}$$

$$\frac{d^2 \log f(\theta)}{(dl_i')^2}\bigg|_{\theta=\hat{\theta}} = -\frac{N - 1 + \alpha}{2} \qquad \frac{d^2 \log f(\theta)}{(dv')^2}\bigg|_{\theta=\hat{\theta}} = -\frac{n(d-k) - 2}{2} \tag{19}$$

The matrix $\mathbf{U}$ is an orthogonal $k$-frame and therefore lives on the Stiefel manifold [7], which is defined by condition (9). The dimension of the manifold is $m = dk - k(k+1)/2$, since we are imposing $k(k+1)/2$ constraints on a $d \times k$ matrix. The prior density for $\mathbf{U}$ is the reciprocal of the area of the manifold [7]:

$$p(\mathbf{U}) = 2^{-k} \prod_{i=1}^k \Gamma((d - i + 1)/2)\pi^{-(d-i+1)/2} \tag{20}$$

A useful parameterization of this manifold is given by the Euler vector representation:

$$\mathbf{U} = \mathbf{U}_d \exp(\mathbf{Z}) \begin{bmatrix} \mathbf{I}_k \\ \mathbf{0} \end{bmatrix} \tag{21}$$

where $\mathbf{U}_d$ is a fixed orthogonal matrix and $\mathbf{Z}$ is a skew-symmetric matrix of parameters, such as

$$\mathbf{Z} = \begin{bmatrix} 0 & z_{12} & z_{13} \\ -z_{12} & 0 & z_{23} \\ -z_{13} & -z_{23} & 0 \end{bmatrix} \tag{22}$$

The first $k$ rows of $\mathbf{Z}$ determine the first $k$ columns of $\exp(\mathbf{Z})$, so the free parameters are $z_{ij}$ with $i < j$ and $i \leq k$; the others are constant. This gives $d(d-1)/2 - (d-k)(d-k-1)/2 = m$ parameters, as desired. For example, in the case $(d = 3, k = 1)$ the free parameters are $z_{12}$ and $z_{13}$, which define a coordinate system for the sphere.

As a function of $\mathbf{U}$, the integrand is simply

$$p(\mathbf{U}|D, \mathbf{L}, v) \propto \exp(-\frac{1}{2}\mathrm{tr}((\mathbf{L}^{-1} - v^{-1}\mathbf{I})\mathbf{U}^\mathsf{T}\mathbf{S}\mathbf{U})) \tag{23}$$

The density is maximized when $\mathbf{U}$ contains the top $k$ eigenvectors of $\mathbf{S}$. However, the density is unchanged if we negate any column of $\mathbf{U}$. This means that there are actually $2^k$ different maxima, and we need to apply Laplace's method to each. Fortunately, these maxima are identical so can simply multiply (16) by $2^k$ to get the integral over the whole manifold. If we set $\mathbf{U}_d$ to the eigenvectors of $\mathbf{S}$:

$$\mathbf{U}_d^\mathsf{T}\mathbf{S}\mathbf{U}_d = N\mathbf{\Lambda} \tag{24}$$

then we just need to apply Laplace's method at $\mathbf{Z} = \mathbf{0}$. As shown in [10], if we define the estimated eigenvalue matrix

$$\hat{\mathbf{\Lambda}} = \begin{bmatrix} \hat{\mathbf{L}} & 0 \\ 0 & \hat{v}\mathbf{I}_{d-k} \end{bmatrix} \tag{25}$$

then the second differential at $\mathbf{Z} = \mathbf{0}$ simplifies to

$$d^2 \log f(\theta)\big|_{\mathbf{Z}=\mathbf{0}} = -\sum_{i=1}^k \sum_{j=i+1}^d (\hat{\lambda}_j^{-1} - \hat{\lambda}_i^{-1})(\lambda_i - \lambda_j)Ndz_{ij}^2 \tag{26}$$

There are no cross derivatives; the Hessian matrix $\mathbf{A}_Z$ is diagonal. So its determinant is the product of these second derivatives:

$$|\mathbf{A}_Z| = \prod_{i=1}^k \prod_{j=i+1}^d (\hat{\lambda}_j^{-1} - \hat{\lambda}_i^{-1})(\lambda_i - \lambda_j)N \tag{27}$$

Laplace's method requires this to be nonsingular, so we must have $k < N$. The cross-derivatives between the parameters are all zero:

$$\left.\frac{d^2 \log f(\theta)}{dl_i d\mathbf{Z}}\right|_{\theta=\hat{\theta}} = \left.\frac{d^2 \log f(\theta)}{dvd\mathbf{Z}}\right|_{\theta=\hat{\theta}} = \left.\frac{d^2 \log f(\theta)}{dl_i dv}\right|_{\theta=\hat{\theta}} = 0 \tag{28}$$

so $\mathbf{A}$ is block diagonal and $|\mathbf{A}| = |\mathbf{A}_Z||\mathbf{A}_L||\mathbf{A}_v|$. We know $\mathbf{A}_L$ from (19), $\mathbf{A}_v$ from (19), and $\mathbf{A}_Z$ from (27). We now have all of the terms needed in (16), and so the evidence approximation is

$$p(D|k) \approx 2^k c_k \left|\hat{\mathbf{L}}\right|^{-n/2} \hat{v}^{-n(d-k)/2} e^{-nd/2} (2\pi)^{(m+k+1)/2} |\mathbf{A}_Z|^{-1/2} |\mathbf{A}_L|^{-1/2} |\mathbf{A}_v|^{-1/2} \tag{29}$$

For model selection, the only terms that matter are those that strongly depend on $k$, and since $\alpha$ is small and $N$ reasonably large we can simplify this to

$$p(D|k) \quad \approx \quad p(\mathbf{U}) \left(\prod_{j=1}^{k} \lambda_j\right)^{-N/2} \hat{v}^{-N(d-k)/2} (2\pi)^{(m+k)/2} |\mathbf{A}_Z|^{-1/2} N^{-k/2} \tag{30}$$

$$\hat{l}_i \quad = \quad \lambda_i \qquad \hat{v} = \frac{\sum_{j=k+1}^{d} \lambda_j}{d-k} \tag{31}$$

which is the recommended formula. Given the eigenvalues, the cost of computing $p(D|k)$ is $O(\min(d, N)k)$, which is less than one loop over the data matrix.

A simplification of Laplace's method is the BIC approximation [8]. This approximation drops all terms which do not grow with $N$, which in this case leaves only

$$p(D|k) \approx \left(\prod_{j=1}^{k} \lambda_j\right)^{-N/2} \hat{v}^{-N(d-k)/2} N^{-(m+k)/2} \tag{32}$$

BIC is compared to Laplace in section 4.

## 4 Results

To test the performance of various algorithms for model selection, we sample data from a known model and see how often the correct dimensionality is recovered. The seven estimators implemented and tested in this study are Laplace's method (30), BIC (32), the two methods of [13] (called RR-N and RR-U), the algorithm in [3] (ER), the ARD algorithm of [1], and 5-fold cross-validation (CV). For cross-validation, the log-probability assigned to the held-out data is the scoring function. ER is the most similar to this paper, since it performs Bayesian model selection on the same model, but uses a different kind of approximation combined with explicit numerical integration. RR-N and RR-U are maximum likelihood techniques on models slightly different than probabilistic PCA; the details are in [10]. ARD is an iterative estimation algorithm for $\mathbf{H}$ which sets columns to zero unless they are supported by the data. The number of nonzero columns at convergence is the estimate of dimensionality.

Most of these estimators work exclusively from the eigenvalues of the sample covariance matrix. The exceptions are RR-U, cross-validation, and ARD; the latter two require diagonalizing a series of different matrices constructed from the data. In our implementation, the algorithms are ordered from fastest to slowest as RR-N, BIC, Laplace, cross-validation, RR-U, ARD, and ER (ER is slowest because of the numerical integrations required).

The first experiment tests the data-rich case where $N >> d$. The data is generated from a 10-dimensional Gaussian distribution with 5 "signal" dimensions and 5 noise dimensions. The eigenvalues of the true covariance matrix are:

| Signal | Noise | |
|---|---|---|
| 10 8 6 4 2 | 1 (×5) | $N = 100$ |

The number of times the correct dimensionality ($k = 5$) was chosen over 60 replications is shown at right. The differences between ER, Laplace, and CV are not statistically significant. Results below the dashed line are worse than Laplace with a significance level of 95%.

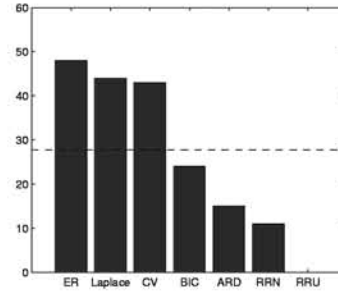

The second experiment tests the case of sparse data and low noise:

| Signal | Noise | |
|---|---|---|
| 10 8 6 4 2 | 0.1 (×10) | $N = 10$ |

The results over 60 replications are shown at right. BIC and ER, which are derived from large $N$ approximations, do poorly. Cross-validation also fails, because it doesn't have enough data to work with.

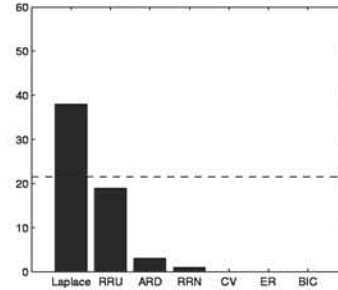

The third experiment tests the case of high noise dimensionality:

| Signal | Noise | |
|---|---|---|
| 10 8 6 4 2 | 0.25 (×95) | $N = 60$ |

The ER algorithm was not run in this case because of its excessive computation time for large $d$.

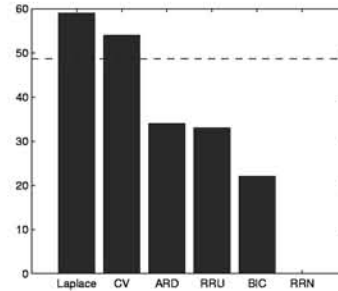

The final experiment tests the robustness to having a non-Gaussian data distribution within the subspace. We start with four sound fragments of 100 samples each. To make things especially non-Gaussian, the values in third fragment are squared and the values in the fourth fragment are cubed. All fragments are standardized to zero mean and unit variance. Gaussian noise in 20 dimensions is added to get:

| Signal | Noise | |
|---|---|---|
| 4 sounds | 0.5 (×20) | $N = 100$ |

The results over 60 replications of the noise (the signals were constant) are reported at right.

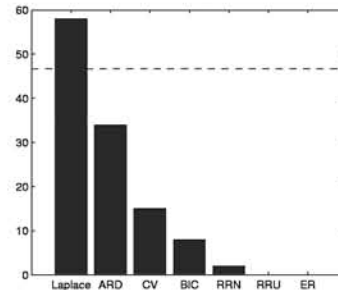

## 5  Discussion

Bayesian model selection has been shown to provide excellent performance when the assumed model is correct or partially correct. The evaluation criterion was the number of times the correct dimensionality was chosen. It would also be useful to evaluate the trained model with respect to its performance on new data within an applied setting. In this case,

Bayesian model averaging is more appropriate, and it is conceivable that a method like ARD, which encompasses a soft blend between different dimensionalities, might perform better by this criterion than selecting one dimensionality.

It is important to remember that these estimators are for density estimation, i.e. accurate representation of the data, and are not necessarily appropriate for other purposes like reducing computation or extracting salient features. For example, on a database of 301 face images the Laplace evidence picked 120 dimensions, which is far more than one would use for feature extraction. (This result also suggests that probabilistic PCA is not a good generative model for face images.)

# References

[1] C. Bishop. Bayesian PCA. In *Neural Information Processing Systems 11*, pages 382–388, 1998.

[2] C. Bregler and S. M. Omohundro. Surface learning with applications to lipreading. In *NIPS*, pages 43–50, 1994.

[3] R. Everson and S. Roberts. Inferring the eigenvalues of covariance matrices from limited, noisy data. *IEEE Trans Signal Processing*, 48(7):2083–2091, 2000.
http://www.robots.ox.ac.uk/~sjrob/Pubs/spectrum.ps.gz.

[4] K. Fukunaga and D. Olsen. An algorithm for finding intrinsic dimensionality of data. *IEEE Trans Computers*, 20(2):176–183, 1971.

[5] Z. Ghahramani and M. Beal. Variational inference for Bayesian mixtures of factor analysers. In *Neural Information Processing Systems 12*, 1999.

[6] Z. Ghahramani and G. Hinton. The EM algorithm for mixtures of factor analyzers. Technical Report CRG-TR-96-1, University of Toronto, 1996.
http://www.gatsby.ucl.ac.uk/~zoubin/papers.html.

[7] A. James. Normal multivariate analysis and the orthogonal group. *Annals of Mathematical Statistics*, 25(1):40–75, 1954.

[8] R. E. Kass and A. E. Raftery. Bayes factors and model uncertainty. Technical Report 254, University of Washington, 1993.
http://www.stat.washington.edu/tech.reports/tr254.ps.

[9] D. J. C. MacKay. Probable networks and plausible predictions — a review of practical Bayesian methods for supervised neural networks. *Network: Computation in Neural Systems*, 6:469–505, 1995.
http://wol.ra.phy.cam.ac.uk/mackay/abstracts/network.html.

[10] T. Minka. Automatic choice of dimensionality for PCA. Technical Report 514, MIT Media Lab Vision and Modeling Group, 1999.
ftp://whitechapel.media.mit.edu/pub/tech-reports/TR-514-ABSTRACT.html.

[11] B. Moghaddam, T. Jebara, and A. Pentland. Bayesian modeling of facial similarity. In *Neural Information Processing Systems 11*, pages 910–916, 1998.

[12] B. Moghaddam and A. Pentland. Probabilistic visual learning for object representation. *IEEE Trans Pattern Analysis and Machine Intelligence*, 19(7):696–710, 1997.

[13] J. J. Rajan and P. J. W. Rayner. Model order selection for the singular value decomposition and the discrete Karhunen-Loéve transform using a Bayesian approach. *IEE Vision, Image and Signal Processing*, 144(2):166–123, 1997.

[14] M. E. Tipping and C. M. Bishop. Mixtures of probabilistic principal component analysers. *Neural Computation*, 11(2):443–482, 1999.
http://citeseer.nj.nec.com/362314.html.

[15] M. E. Tipping and C. M. Bishop. Probabilistic principal component analysis. *J Royal Statistical Society B*, 61(3), 1999.
